# Practical Characteristics of Neural Network and Conventional Pattern Classifiers on Artificial and Speech Problems*

**Yuchun Lee**
Digital Equipment Corp.
40 Old Bolton Road,
OGO1-2U11
Stow, MA 01775-1215

**Richard P. Lippmann**
Lincoln Laboratory, MIT
Room B-349
Lexington, MA 02173-9108

## ABSTRACT

Eight neural net and conventional pattern classifiers (Bayesian-unimodal Gaussian, k-nearest neighbor, standard back-propagation, adaptive-stepsize back-propagation, hypersphere, feature-map, learning vector quantizer, and binary decision tree) were implemented on a serial computer and compared using two speech recognition and two artificial tasks. Error rates were statistically equivalent on almost all tasks, but classifiers differed by orders of magnitude in memory requirements, training time, classification time, and ease of adaptivity. Nearest-neighbor classifiers trained rapidly but required the most memory. Tree classifiers provided rapid classification but were complex to adapt. Back-propagation classifiers typically required long training times and had intermediate memory requirements. These results suggest that classifier selection should often depend more heavily on practical considerations concerning memory and computation resources, and restrictions on training and classification times than on error rate.

*This work was sponsored by the Department of the Air Force and the Air Force Office of Scientific Research.

# 1    Introduction

A shortcoming of much recent neural network pattern classification research has been an overemphasis on back-propagation classifiers and a focus on classification error rate as the main measure of performance. This research often ignores the many alternative classifiers that have been developed (see e.g. [10]) and the practical tradeoffs these classifiers provide in training time, memory requirements, classification time, complexity, and adaptivity. The purpose of this research was to explore these tradeoffs and gain experience with many different classifiers. Eight neural net and conventional pattern classifiers were used. These included Bayesian-unimodal Gaussian, k-nearest neighbor (kNN), standard back-propagation, adaptive-stepsize back-propagation, hypersphere, feature-map (FM), learning vector quantizer (LVQ), and binary decision tree classifiers.

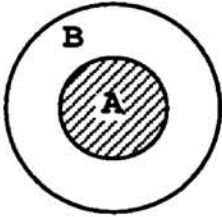

BULLSEYE

Dimensionality: 2
Testing Set Size: 500
Training Set Size: 500
Classes: 2

DISJOINT

Dimensionality: 2
Testing Set Size: 500
Training Set Size: 500
Classes: 2

DIGIT

Dimensionality: 22 Cepstra
Training Set Size: 70
Testing Set Size: 112
16 Training Sets
16 Testing Sets
Classes:  7 Digits
Talker Dependent

VOWEL

Dimension: 2 Formants
Training Set Size: 338
Testing Set Size: 330
Classes:  10 Vowels
Talker Independent

**Figure 1:** Four problems used to test classifiers.

Classifiers were implemented on a serial computer and tested using the four problems shown in Fig. 1. The upper two artificial problems (Bullseye and Disjoint) require simple two-dimensional convex or disjoint decision regions for minimum error classification. The lower digit recognition task (7 digits, 22 cepstral parameters,

16 talkers, 70 training and 112 testing patterns per talker) and vowel recognition task (10 vowels, 2 formant parameters, 67 talkers, 338 training and 330 testing patterns) use real speech data and require more complex decision regions. These tasks are described in [6, 11] and details of experiments are available in [9].

## 2    Training and Classification Parameter Selection

Initial experiments were performed to select sizes of classifiers that provided good performance with limited training data and also to select high-performing versions of each type of classifier. Experiments determined the number of nodes and hidden layers in back-propagation classifiers, pruning techniques to use with tree and hypersphere classifiers, and numbers of exemplars or kernel nodes to use with feature-map and LVQ classifiers.

### 2.1    Back-Propagation Classifiers

In standard back-propagation, weights typically are updated only after each *trial* or *cycle*. A trial is defined as a single training pattern presentation and a cycle is defined as a sequence of trials which sample all patterns in the training set. In group updating, weights are updated every $T$ trials while in trial-by-trial training, weights are updated every trial. Furthermore, in trial-by-trial updating, training patterns can be presented *sequentially* where a pattern is guaranteed to be presented every $T$ trials, or they can be presented *randomly* where patterns are randomly selected from the training set. Initial experiments demonstrated that random trial-by-trial training provided the best convergence rate and error reduction during training. It was thus used whenever possible with all back-propagation classifiers.

All back-propagation classifiers used a single hidden layer and an output layer with as many nodes as classes. The classification decision corresponded to the class of the node in the output layer with the highest output value. During training, the desired output pattern, $D$, was a vector with all elements set to 0 except for the element corresponding to the correct class of the input pattern. This element of $D$ was set to 1. The mean-square difference between the actual output and this desired output error is minimized when the output of each node is exactly the Bayes *a posteriori* probability for each correct class [1, 10]. Back-propagation with this "1 of $m$" desired output is thus well justified theoretically because it attempts to estimate minimum-error Bayes probability functions. The number of hidden nodes used in each back-propagation classifier was determined experimentally as described in [6, 7, 9, 11].

Three "improved" back-propagation classifiers with the potential of reduced training times where studied. The first, the *adaptive-stepsize-classifier*, has a global stepsize that is adjusted after every training cycle as described in [4]. The second, the *multiple-adaptive-stepsize classifier*, has multiple stepsizes (one for each weight) which are adjusted after every training cycle as described in [8]. The third classifier uses the conjugate gradient method [9, 12] to minimize the output mean-square error.

The goal of the three "improved" versions of back-propagation was to shorten the often lengthy training time observed with standard back-propagation. These improvements relied on fundamental assumptions about the error surfaces. However, only the multiple-adaptive-stepsize algorithm was used for the final classifier comparison due to the poor performance of the other two algorithms. The adaptive-stepsize classifier often could not achieve adequately low error rates because the global stepsize ($\eta$) frequently converged too quickly to zero during training. The multiple-adaptive-stepsize classifier did not train faster than a standard back-propagation classifier with carefully selected stepsize value. Nevertheless, it eliminated the need for pre-selecting the stepsize parameter. The conjugate gradient classifier worked well on simple problems but almost always rapidly converged to a local minimum which provided high error rates on the more complex speech problems.

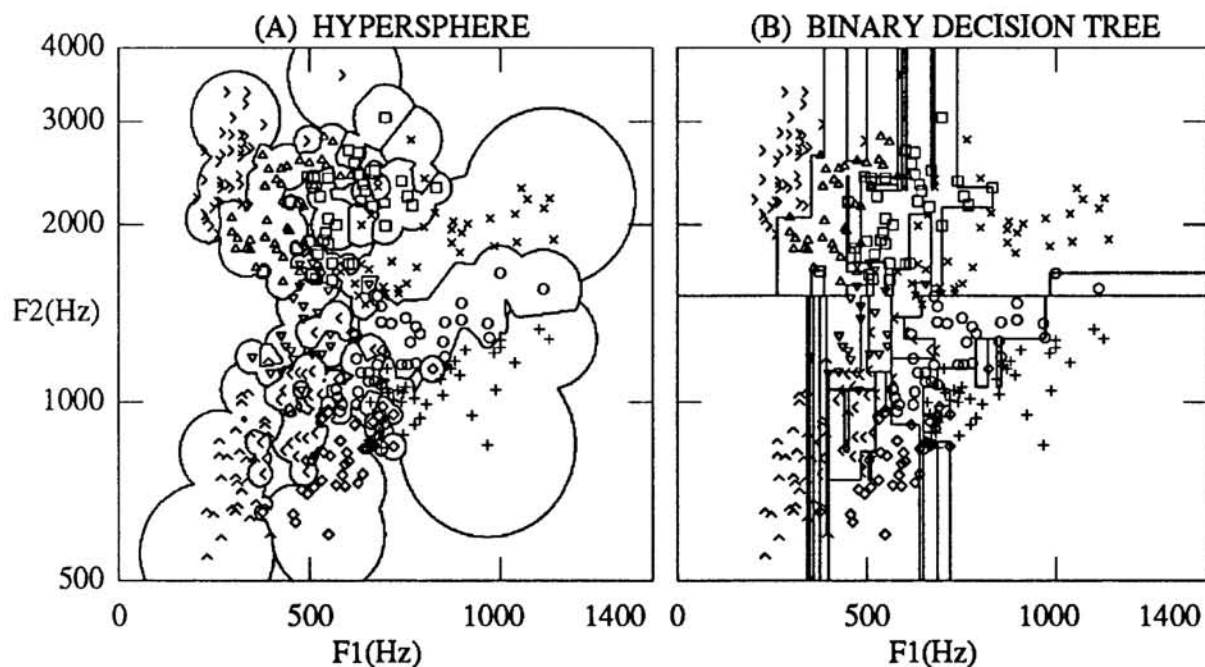

**Figure 2:** Decision regions formed by the hypersphere classifier (A) and by the binary decision tree classifier (B) on the test set for the vowel problem. Inputs consist of the first two formants for ten vowels in the words ∧ who'd, ◇ hawed, + hod, ○ hud, × had, > heed, △ hid, □ head, ▽ heard, and < hood as described in [6, 9].

## 2.2  Hypersphere Classifier

Hypersphere classifiers build decision regions from nodes that form separate hypersphere decision regions. Many different types of hypersphere classifiers have been developed [2, 13]. Experiments discussed in [9], led to the selection of a specific version of hypersphere classifier with "pruning". Each hypersphere can only shrink in size, centers are not repositioned, an ambiguous response (positive outputs from hyperspheres corresponding to different classes) is mediated using a nearest-neighbor

rule, and hyperspheres that do not contribute to the classification performance are pruned from the classifier for proper "fitting" of the data and to reduce memory usage. Decision regions formed by a hypersphere classifier for the vowel classification problem are shown in the left side of Fig. 2. Separate regions in this figure correspond to different vowels. Decision region boundaries contain arcs which are segments of hyperspheres (circles in two dimensions) and linear segments caused by the application of the nearest neighbor rule for ambiguous responses.

### 2.3   Binary Decision Tree Classifier

Binary decision tree classifiers from [3] were used in all experiments. Each node in a tree has only two immediate offspring and the splitting decision is based on only one of the input dimensions. Decision boundaries are thus overlapping *hyper-rectangles* with sides parallel to the axes of the input space and decision regions become more complex as more nodes are added to the tree. Decision trees for each problem were grown until they classified all the training data exactly and then pruned back using the test data to determine when to stop pruning. A complete description of the decision tree classifier used is provided in [9] and decision regions formed by this classifier for the vowel problem are shown in the right side of Fig. 2.

### 2.4   Other Classifiers

The remaining four classifiers were tuned by selecting coarse sizing parameters to "fit" the problem imposed. Some of these parameters include the number of exemplars in the LVQ and feature map classifiers and $k$ in the $k$-nearest neighbor classifier. Different types of covariance matrices (full, diagonal, and various types of grand averaging) were also tried for the Bayesian-unimodal Gaussian classifier. Best sizing parameter values for classifiers were almost always not those that that best classified the training set. For the purpose of this study, training data was used to determine internal parameters or weights in classifiers. The size of a classifier and coarse sizing parameters were selected using the test data. In real applications when a test set is not available, alternative methods, such as *cross validation*[3, 14] would be used.

## 3   Classifier Comparison

All eight classifiers were evaluated on the four problems using simulations programmed in C on a Sun 3/110 workstation with a floating point accelerator. Classifiers were trained until their training error rate converged.

### 3.1   Error Rates

Error rates for all classifiers on all problems are shown in Fig. 3. The middle solid lines in this figure correspond to the average error rate over all classifiers for each problem. The shaded area is one binomial standard deviation above and below this average. As can be seen, there are only three cases where the error rate of any one classifier is substantially different from the average error. These exceptions are the Bayesian-unimodal Gaussian classifier on the disjoint problem

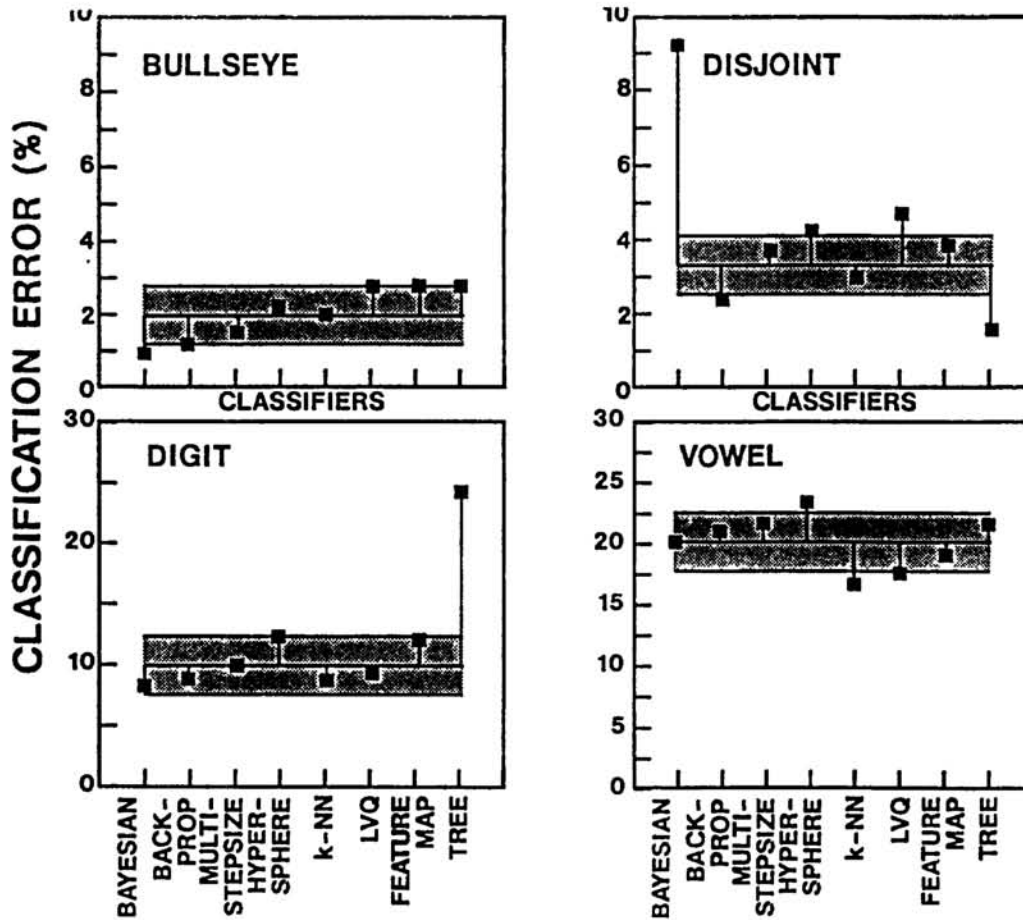

**Figure 3:** Error rates for all classifiers on all four problems. The middle solid lines correspond to the average error rate over all classifiers for each problem. The shaded area is one binomial standard deviation above and below the average error rate.

and the decision tree classifier on the digit and the disjoint problem. The Bayesian-unimodal Gaussian classifier performed poorly on the disjoint problem because it was unable to form the required bimodal disjoint decision regions. The decision tree classifier performed poorly on the digit problem because the small amount of training data (10 patterns per class) was adequately classified by a minimal 13-node tree which didn't generalize well and didn't even use all 22 input dimensions. The decision tree classifier worked well for the disjoint problem because it forms decision regions parallel to both input axes as required for this problem.

## 3.2   Practical Characteristics

In contrast to the small differences in error rate, differences between classifiers on practical performance issues such as training and classification time, and memory usage were large. Figure 4 shows that the classifiers differed by orders of magnitude in training time. Shown in log-scale, the k-nearest neighbor stands out distinctively

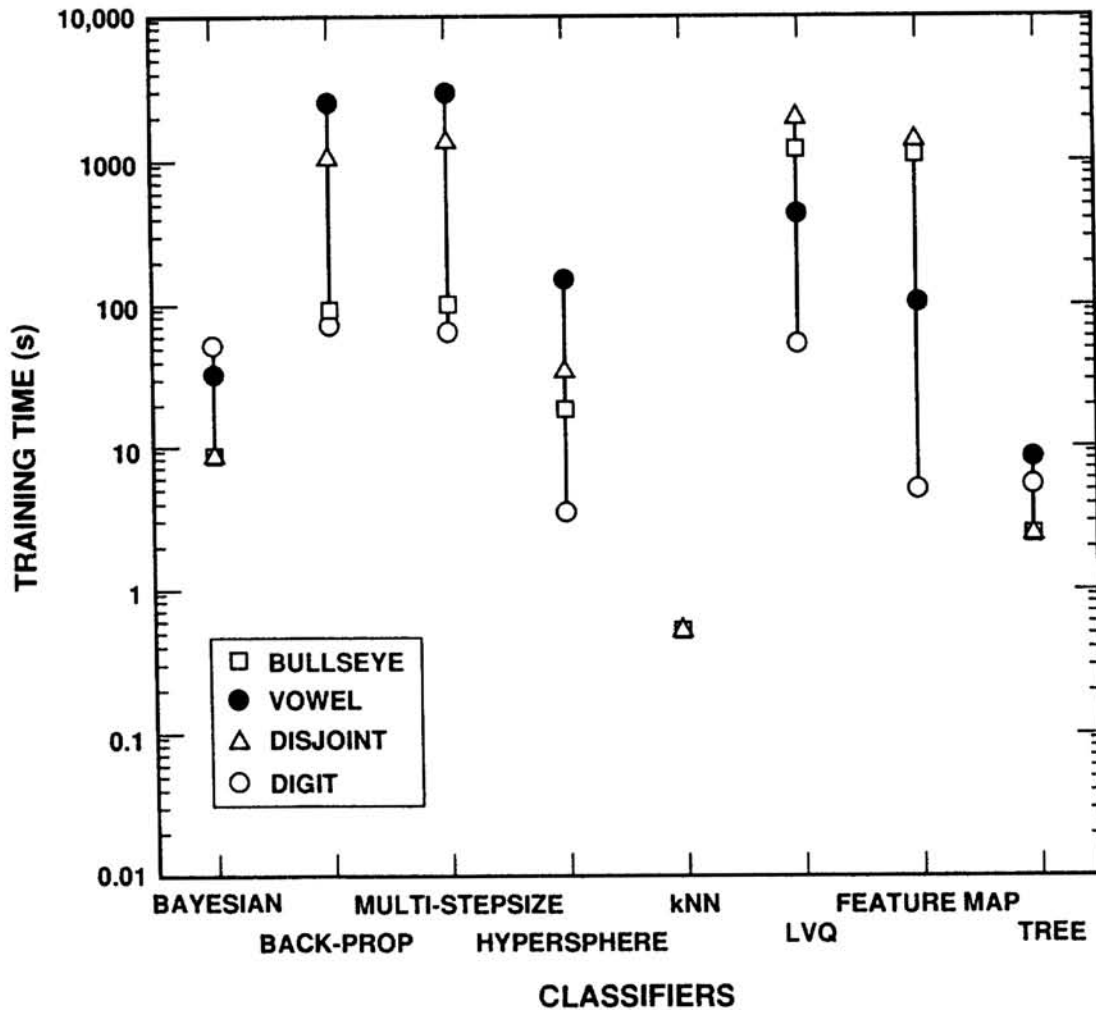

**Figure 4:** Training time of all classifiers on all four problems.

as the fastest trained classifier by many orders of magnitude. Depending on the problem, Bayesian-unimodal Gaussian, hypersphere, decision tree, and feature map classifiers also have reasonably short training times. LVQ and back-propagation classifiers often required the longest training time. It should be noted that alternative implementations, for example using parallel computers, would lead to different results.

Adaptivity or the ability to adapt using new patterns after complete training also differed across classifiers. The k-nearest neighbor and hypersphere classifiers are able to incorporate new information most readily. Others such as back-propagation and LVQ classifiers are more difficult to adapt and some, such as decision tree classifiers, are not designed to handle further adaptation after training is complete.

The binary decision tree can classify patterns much faster than others. Unlike most classifiers that depend on "distance" calculations between the input pattern and all stored exemplars, the decision tree classifier requires only a few numerical comparisons. Therefore, the decision tree classifier was many orders of magnitude faster

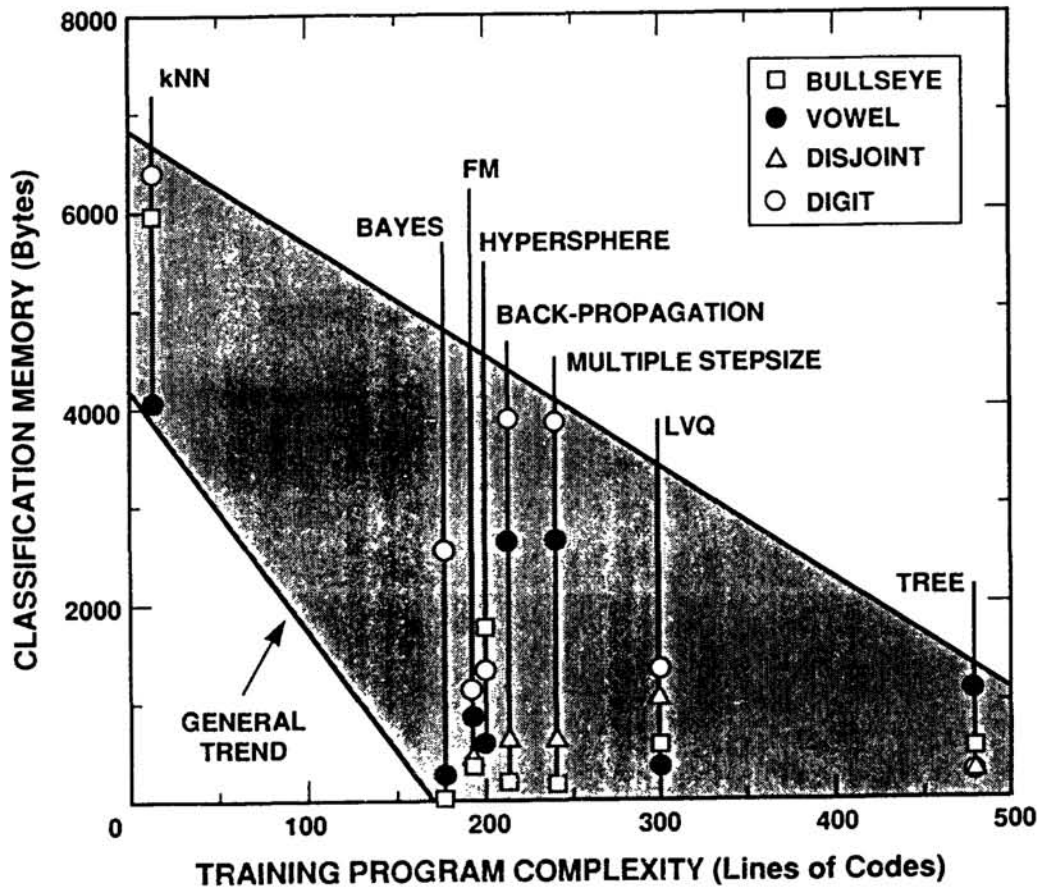

**Figure 5:** Classification memory usage versus training program complexity for all classifiers on all four problems.

in classification than other classifiers. However, decision tree classifiers require the most complex training algorithm. As a rough measurement of the ease of implementation, subjectively measured by the number of lines in the training program, the decision tree classifier is many times more complex than the simplest training program– that of the *k*-nearest neighbor classifier. However, the *k*-nearest neighbor classifier is one of the slowest in classification when implemented serially without complex search techniques such as k-d trees [5]. These techniques greatly reduce classification time but make adaptation to new training data more difficult and increase complexity.

## 4   Trade-Offs Between Performance Criteria

No one classifier out-performed the rest on all performance criteria. The selection of a "best" classifier depends on practical problem constraints which differ across problems. Without knowing these constraints or associating explicit costs with various performance criteria, a classifier that is "best" can not be meaningfully determined. Instead, there are numerous trade-off relationships between various criteria.

One trade-off shown in Fig. 5 is classification memory usage versus the complexity of the training algorithm. The far upper left corner, where training is very simple and memory is not efficiently utilized, contains the *k*-nearest neighbor classifier. In contrast, the binary decision tree classifier is in the lower right corner, where the overall memory usage is minimized and the training process is very complex. Other classifiers are intermediate.

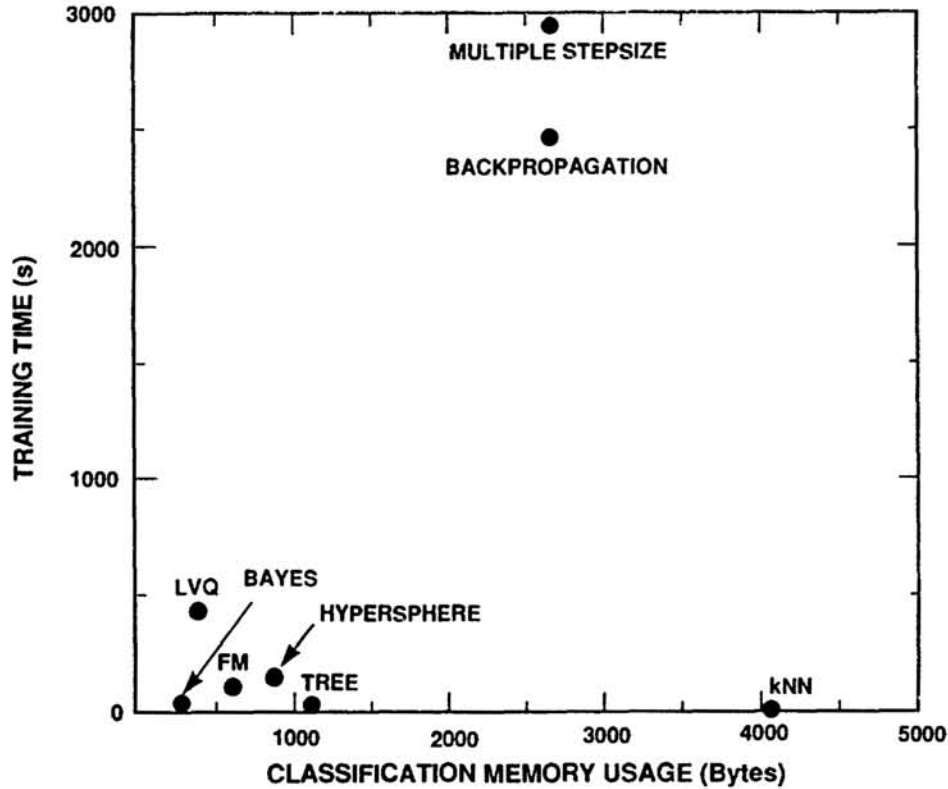

**Figure 6:** Training time versus classification memory usage of all classifiers on the vowel problem.

Figure 6 shows the relationship between training time and classification memory usage for the vowel problem. The *k*-nearest neighbor classifier consistently provides the shortest training time but requires the most memory. The hypersphere classifier optimizes these two criteria well across all four problems. Back-propagation classifiers frequently require long training times and require intermediate amounts of memory.

## 5    Summary

This study explored practical characteristics of neural net and conventional pattern classifiers. Results demonstrate that classification error rates can be equivalent across classifiers when classifiers are powerful enough to form minimum error decision regions, when they are rigorously tuned, and when sufficient training data is provided. Practical characteristics such as training time, memory requirements, and classification time, however, differed by orders of magnitude. In practice, these factors are more likely to affect classifier selection. Selection will often be driven

by practical considerations concerning memory and computation resources, restrictions on training, test, and adaptation times, and ease of use and implementation. The many existing neural net and conventional classifiers allow system designers to trade these characteristics off. Tradeoffs will vary with implementation hardware (e.g. serial versus parallel, analog versus digital ) and details of the problem (e.g. dimension of the input vector, complexity of decision regions). Our current research efforts are exploring these tradeoffs on more difficult problems and studying additional classifiers including radial-basis-function classifiers, high-order networks, and Gaussian mixture classifiers.

# References

[1] A. R. Barron and R. L. Barron. Statistical learning networks: A unifying view. In *1988 Symposium on the Interface: Statistics and Computing Science*, Reston, Virginia, April 21-23 1988.

[2] B. G. Batchelor. Classification and data analysis in vector space. In B. G. Batchelor, editor, *Pattern Recognition*, chapter 4, pages 67–116. Plenum Press, London, 1978.

[3] L. Breiman, J. H. Friedman, R. A. Olshen, and C. J. Stone. *Classification and Regression Trees*. Wadsworth International Group, Belmont, CA, 1984.

[4] L. W. Chan and F. Fallside. An adaptive training algorithm for back propagation networks. *Computer Speech and Language*, 2:205–218, 1987.

[5] J. H. Friedman, J. L. Bentley, and R. A. Finkel. An algorithm for finding best matches in logarithmic expected time. *ACM Transactions on Mathematical Software*, 3(3):209–226, September 1977.

[6] W. M. Huang and R. P. Lippmann. Neural net and traditional classifiers. In D. Anderson, editor, *Neural Information Processing Systems*, pages 387–396, New York, 1988. American Institute of Physics.

[7] William Y. Huang and Richard P. Lippmann. Comparisons between conventional and neural net classifiers. In *1st International Conference on Neural Networks*, pages IV–485. IEEE, June 1987.

[8] R. A. Jacobs. Increased rates of convergence through learning rate adaptation. *Neural Networks*, 1:295–307, 1988.

[9] Yuchun Lee. Classifiers: Adaptive modules in pattern recognition systems. Master's thesis, Massachusetts Institute of Technology, Department of Electrical Engineering and Computer Science, Cambridge, MA, May 1989.

[10] R. P. Lippmann. Pattern classification using neural networks. *IEEE Communications Magazine*, 27(11):47–54, November 1989.

[11] Richard P. Lippmann and Ben Gold. Neural classifiers useful for speech recognition. In *1st International Conference on Neural Networks*, pages IV–417. IEEE, June 1987.

[12] W. H. Press, B. P. Flannery, S. A. Teukolsky, and W. T. Vetterling, editors. *Numerical Recipes*. Cambridge University Press, New York, 1986.

[13] D. L. Reilly, L. N. Cooper, and C. Elbaum. A neural model for category learning. *Biological Cybernetics*, 45:35–41, 1982.

[14] M. Stone. Cross-validation choice and assessment of statistical predictions. *Journal of the Royal Statistical Society*, B-36:111–147, 1974.
